# Large-scale biophysical parameter estimation in single neurons via constrained linear regression

**Misha B. Ahrens**[*], **Quentin J.M. Huys**[*], **Liam Paninski**

Gatsby Computational Neuroscience Unit
University College London
{ahrens, qhuys, liam}@gatsby.ucl.ac.uk

## Abstract

Our understanding of the input-output function of single cells has been substantially advanced by biophysically accurate multi-compartmental models. The large number of parameters needing hand tuning in these models has, however, somewhat hampered their applicability and interpretability. Here we propose a simple and well-founded method for automatic estimation of many of these key parameters: 1) the spatial distribution of channel densities on the cell's membrane; 2) the spatiotemporal pattern of synaptic input; 3) the channels' reversal potentials; 4) the intercompartmental conductances; and 5) the noise level in each compartment. We assume experimental access to: a) the spatiotemporal voltage signal in the dendrite (or some contiguous subpart thereof, e.g. via voltage sensitive imaging techniques), b) an approximate kinetic description of the channels and synapses present in each compartment, and c) the morphology of the part of the neuron under investigation. The key observation is that, given data a)-c), all of the parameters 1)-4) may be simultaneously inferred by a version of constrained linear regression; this regression, in turn, is efficiently solved using standard algorithms, without any "local minima" problems despite the large number of parameters and complex dynamics. The noise level 5) may also be estimated by standard techniques. We demonstrate the method's accuracy on several model datasets, and describe techniques for quantifying the uncertainty in our estimates.

## 1   Introduction

The usual tradeoff in parameter estimation for single neuron models is between realism and tractability. Typically, the more biophysical accuracy one tries to inject into the model, the harder the computational problem of fitting the model's parameters becomes, as the number of (nonlinearly interacting) parameters increases (sometimes even into the thousands, in the case of complex multicompartmental models).

---
[*]These authors contributed equally. Support contributed by the Gatsby Charitable Foundation (LP, MA), a Royal Society International Fellowship (LP), the BIBA consortium and the UCL School of Medicine (QH). We are indebted to P. Dayan, M. Häusser, M. London, A. Roth, and S. Roweis for helpful and interesting discussions, and to R. Wood for channel definitions.

Previous authors have noted the difficulties of this large-scale, simultaneous parameter estimation problem, which are due both to the highly nonlinear nature of the "cost functions" minimized (e.g., the percentage of correctly-predicted spike times [1]) and the abundance of local minima on the very large-dimensional allowed parameter space [2, 3].

Here we present a method that is both computationally tractable and biophysically detailed. Our goal is to simultaneously infer the following dendritic parameters: 1) the spatial distribution of channel densities on the cell's membrane; 2) the spatiotemporal pattern of synaptic input; 3) the channels' reversal potentials; 4) the intercompartmental conductances; and 5) the noise level in each compartment. Achieving this somewhat ambitious goal comes at a price: our method assumes that the experimenter a) knows the geometry of the cell, b) has a good understanding of the kinetics of the channels present in each compartment, and c) most importantly, is able to observe the spatiotemporal voltage signal on the dendritic tree, or at least a fraction thereof (e.g. by voltage-sensitive imaging methods; in electrotonically compact cells, single electrode recordings can be used).

The key to the proposed method is to recognise that, when we condition on data a)-c), the dynamics governing this observed spatiotemporal voltage signal become *linear* in the parameters we are seeking to estimate (even though the system itself may behave highly nonlinearly), so that the parameter estimation can be recast into a simple constrained linear regression problem (see also [4, 5]). This implies, somewhat counterintuitively, that optimizing the likelihood of the parameters in this setting is a *convex* problem, with no non-global local extrema. Moreover, linearly constrained quadratic optimization is an extremely well-studied problem, with many efficient algorithms available. We give examples of the resulting methods successfully applied to several types of model data below. In addition, we discuss methods for incorporating prior knowledge and analyzing uncertainty in our estimates, again basing our techniques on the well-founded probabilistic regression framework.

## 2 Methods

Biophysically accurate models of single cells are typically formulated compartmentally – a set of first-order coupled differential equations that form a spatially discrete approximation to the cable equations. Modeling the cell under investigation in this discretized manner, a typical equation describing the voltage in compartment $x$ is

$$C_x dV_x(t) = \left( \sum_i a_{i,x} J_{i,x}(t) + I_x(t) \right) dt + \sigma_x dN_{x,t}. \tag{1}$$

Here $\sigma_x N_{x,t}$ is evolution (current) noise and $I_x(t)$ is externally injected current. Dropping the subscript $x$ where possible, the terms $a_i \cdot J_i(t)$ represent currents due to:

1. voltage mismatch in neighbouring compartments, $f_{x,y}(V_y(t) - V_x(t))$,
2. synaptic input, $g_s(t)(E_s - V(t))$,
3. membrane channels, active (voltage-dependent) or passive, $\bar{g}_j g_j(t)(E_j - V(t))$.

Here $a_i$ are parameters to be inferred:

1. the intercompartmental conductances $f_{x,y}$,
2. the spatiotemporal input from synapse $s$, $u_s(t)$, from which $g_s(t)$ is obtained by

$$dg_s(t)/dt = -g_s(t)/\tau_s + u_s(t), \tag{2}$$

a linear convolution operation (the synaptic kinetic parameter $\tau_s$ is assumed known) which may be written in matrix notation $\mathbf{g}_s = \mathbf{K}\mathbf{u}$.

3. the ion channel concentrations $\bar{g}_j$. The open probabilities of channel $j$, $g_j(t)$, are obtained from the *channel kinetics*, which are assumed to evolve deterministically, with a known dependence on $V$, as in the Hodgkin-Huxley model, $g_{Na} = m^3 h$,

$$\tau_m dm(t)/dt = m_\infty(V) - m, \qquad (3)$$

and similarly for $h$. Again, we emphasize that the kinetic parameters $\tau_m$ and $m_\infty(V)$ are assumed known; only the inhomogeneous concentrations are unknown. (For passive channels $g_j$ is taken constant and independent of voltage.)

The parameters 1-3 are relative to membrane capacitance $C_x$.[1]

When modeling the dynamics of a single neuron according to (1), the voltage $V(t)$ and channel kinetics $g_j(t)$ are typically evolved in parallel, according to the injected current $I(t)$ and synaptic inputs $u_s(t)$. Suppose, on the other hand, that we have observed the voltage $V_x(t)$ in each compartment. Since we have assumed we also know the channel kinetics (equation 3), the synaptic kinetics (equation 2) and the reversal potentials $E_j$ of the channels present in each compartment, we may decouple the equations and determine the open probabilities $g_{j,x}(t)$ for $t \in [0, T]$. This, in turn, implies that the currents $J_{i,x}(t)$ and voltage differentials $\dot{V}_x(t)$ are all known, and we may interpret equation 1 as a *regression equation*, linear in the unknown parameters $a_i$, instead of an evolution equation. This is the key observation of this work.

Thus we can use linear regression methods to simultaneously infer optimal values of the parameters $\{\bar{g}_{j,x}, u_{s,x}(t), f_{x,y}\}$[2]. More precisely, rewrite equation (1) in matrix form, $\dot{\mathbf{V}} = \mathbf{Ma} + \sigma\eta$, where each column of the matrix $\mathbf{M}$ is composed of one of the known currents $\{J_i(t), t \in [0, T]\}$ (with $T$ the length of the experiment) and the column vectors $\dot{\mathbf{V}}$, $\mathbf{a}$, and $\eta$ are defined in the obvious way. Then

$$\hat{\mathbf{a}}_{opt} = \arg\min_{\mathbf{a}} \|\dot{\mathbf{V}} - \mathbf{Ma}\|_2^2. \qquad (4)$$

In addition, since on physical grounds the channel concentrations, synaptic input, and conductances must be non-negative, we require our solution $a_i \geq 0$. The resulting linearly-constrained quadratic optimization problem has no local minima (due to the convexity of the objective function and of the domain $g_i \geq 0$), and allows quadratic programming (QP) tools (e.g., quadprog.m in Matlab) to be employed for highly efficient optimization.

**Quadratic programming tactics**: As emphasized above, the dimension $d$ of the parameter space to be optimized over in this application is quite large ($d \sim N_{comp}(TN_{syn} + N_{chan})$, with $N$ denoting the number of compartments, synapse types, and membrane channel types respectively). While our problem is convex, and therefore tractable in the sense of having no nonglobal local optima, the time-complexity of QP, implemented naively, is $\mathcal{O}(d^3)$, which is too slow for our purposes.

Fortunately, the correlational structure of the parameters allows us to perform this optimization more efficiently, by several natural decompositions: in particular, given the spatiotemporal voltage signal $V_x(t)$, parameters which are distant in space (e.g., the densities of channels in widely-separated compartments) and time (i.e., the synaptic input $u_{s,x}(t)$ for $t = t_i$ and $t_j$ with $|t_i - t_j|$ large) may be optimized independently. This amounts to a kind of "coordinate descent" algorithm, in which we decompose our parameter set into a set of (not necessarily disjoint) subsets, and iteratively optimize the parameters in each subset

while holding all the other parameters fixed. (The quadratic nature of the original problem guarantees that each of these subset problems will be quadratic, with no local minima.) Empirically, we found that this decomposition / sequential optimization approach reduced the computation time from $\mathcal{O}(d^3)$ to near $\mathcal{O}(d)$.

## 2.1 The probabilistic framework

If we assume the noise $N_{x,t}$ is Gaussian and white, then the mean-square regression solution for $\mathbf{a}$ described above coincides exactly with the (constrained) maximum likelihood estimate, $\hat{\mathbf{a}}_{ML} = \arg\min_{\mathbf{a}} \|\dot{\mathbf{V}} - \mathbf{Ma}\|_2^2 / 2\sigma^2$. (The noise scale $\sigma$ may also be estimated via maximum likelihood.) This suggests several straightforward likelihood-based techniques for representing the uncertainty in our estimates.

**Posterior confidence intervals**: The assumption of Gaussian noise implies that the posterior distribution of the parameters $\mathbf{a}$ is of the form $p(\mathbf{a}|\mathbf{V}) = \frac{1}{Z}p(\mathbf{a})G_{\mu,\Sigma}(\mathbf{a})$, with $Z$ a normalizing constant, the prior $p(\mathbf{a})$ supported on $a_i \geq 0$, and the mean and covariance of the likelihood Gaussian $G(\mathbf{a})$ given by $\mu = (\mathbf{M}^T\mathbf{M})^{-1}\mathbf{M}^T\dot{\mathbf{V}}$ and $\Sigma^{-1} = \mathbf{M}^T\mathbf{M}/\sigma^2$. We will assume a flat prior distribution $p(\mathbf{a})$ (that is, no prior knowledge) on the non-synaptic parameters $\{\bar{g}_{j,x}, f_{x,y}\}$ (although clearly non-flat priors can be easily incorporated here [6]); for the synaptic parameters $u_{s,x}(t)$ it will be convenient to use a product-of-exponentials prior, $p(\mathbf{u}) = \prod_i \lambda_i \exp(-\lambda_i u_i)$. In each case, computing confidence intervals for $a_i$ reduces to computing moments of multidimensional Gaussian distributions, truncated to $a_i \geq 0$.

We use importance sampling methods [7] to compute these moments for the channel parameters. Sampling from high-dimensional truncated Gaussians via sample-reject is inefficient (since samples from the non-truncated Gaussian – call this distribution $p^*(\mathbf{a}|\mathbf{V})$ – may violate the constraint $a_i \geq 0$ with high probability). Therefore we sample instead from a proposal density $q(\mathbf{a})$ with support on $a_i \geq 0$ (specifically, a product of univariate truncated Gaussians with mean $\mathbf{a}_i$ and appropriate variance) and evaluate the second moments around $\mathbf{a}_{ML}$ by

$$\mathbb{E}[(a_i - a_{MLi})^2|\mathbf{V}] \approx \frac{1}{Z}\sum_{n=1}^{N}\frac{p^*(\mathbf{a}^n|\mathbf{V})}{q(\mathbf{a}^n)}(a_i^n - a_{MLi})^2 \qquad \text{where} \qquad Z = \sum_{n=1}^{N}\frac{p^*(\mathbf{a}^n|\mathbf{V})}{q(\mathbf{a}^n)}.$$

(5)

**Hessian Principal Components Analysis**: The procedure described above allows us to quantify the uncertainty of individual estimated parameters $a_i$. We are also interested in the uncertainty of our estimates in a joint sense (e.g., in the posterior covariance instead of just the individual variances). The negative Hessian of the loglikelihood function, $\mathbf{A} \sim \mathbf{M}^T\mathbf{M}$, contains a great deal of this information, which may be extracted via a kind of principal components analysis: the eigenvectors of $\mathbf{A}$ corresponding to the greatest eigenvalues tell us in which directions the model is most strongly constrained by the data, while low eigenvalues correspond to directions in which the likelihood changes relatively slowly, e.g. channels whose corresponding currents are highly correlated (and therefore approximately interchangeable). These ideas will be illustrated in section 3.4.

## 3   Results

To test the validity, efficiency and accuracy of the proposed method we apply it to model data of varying complexity.

### 3.1 Inferring channel conductances in a multicompartmental model

We take a simple 14-compartment model neuron, described by

$$C_x \frac{dV_x}{dt} = \sum_{c=1}^{N_{chan}} \bar{g}_c g_c(V_x, t)(E_c - V_x(t)) + \sum_y f_{x,y} \cdot (V_y(t) - V_x(t)) + I_x(t) + \sigma_x dN_{x,t};$$

recall $f_{x,y}$ are the intercompartmental conductances, $g_c(V, t)$ is channel $c$'s conductance state given the voltage history up to time $t$, and $\bar{g}_c$ is the channel concentration. We minimize a vectorized expression as above (equation 4). On biophysical grounds we require $f_{x,y} = f_{y,x}$; we enforce this (linear) constraint by only including one parameter for each connected pair of compartments $(x, y)$. In this case the true channel kinetics were of standard Hodgkin-Huxley form (Na$^+$, K$^+$ and leak), with inhomogeneous densities (figure 1). To test the selectivity of the estimation procedure, we fitted $N_{chan} = 8$ candidate channels from [8, 9, 10] (five of which were absent in the true model cell). Figure 1 shows the performance of the inference; despite the fact that we used only 20 ms of model data, the last 7 ms of which were used for the actual fitting (the first 13 ms were used to evolve the random initial conditions to an approximately correct value), the fit is near perfect in the $\sigma = 0$ case, with vanishingly small errorbars. The concentrations of the five channels that were not present when generating the data were set to approximately zero, as desired (data not shown). The lower panels demonstrate the robustness of the methods on highly noisy (large $\sigma$) data, in which case the estimated errorbars become significant, but the performance degrades only slightly.

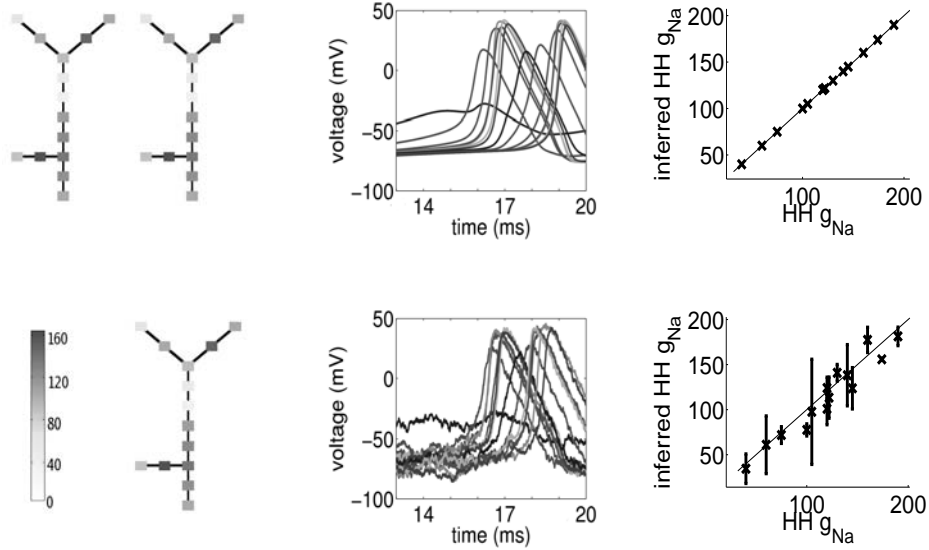

**Figure 1**: Top panels: $\sigma = 0$. 14 compartment model neuron, Na$^+$ channel concentration indicated by grey scale; estimated Na$^+$ channel concentrations in the noiseless case; observed voltage traces (one per compartment); estimated concentrations. Bottom panels: $\sigma$ large. Na$^+$ channel concentration legend, values relative to $C_m$ (e.g. in $mS/cm^2$ if $C_m = 1\mu F/cm^2$); estimated Na$^+$ concentrations in the noisy case; noisy voltage traces; estimated channel concentrations. K$^+$ channel concentrations and intercompartmental conductances $f_{x,y}$ not shown (similar performance).

### 3.2 Inferring synaptic input in a passive model

Next we simulated a single-compartment, leaky neuron (i.e., no voltage-sensitive membrane channels) with synaptic input from three synapses, two excitatory (glutamatertic;

$\tau = 3$ ms, $E = 0$ mV) and one inhibitory (GABA$_A$; $\tau = 5$ ms, $E = -75$ mV). When we attempted to estimate the synaptic input $u_s(t)$ via the ML estimator described above (figure 2, left), we observe an *overfitting* phenomenon: the current noise due to $N_t$ is being "explained" by competing balanced excitatory and inhibitory synaptic inputs. This overfitting is unsurprising, given that we are modeling a $T$-dimensional observation, $\dot{\mathbf{V}}$, with $2T$ regressor variables, $u_-(t)$ and $u_+(t), 0 < t < T$ (indeed, overfitting is much less apparent in the case that only one synapse is modeled, where no balance of excitation and inhibition is possible; data not shown).

Once again, we may make use of well-known techniques from the regression literature to solve this problem: in this case, we need to regularize our estimated synaptic parameters. Instead of maximizing the likelihood, $\mathbf{u}_{ML}$, we maximize the *posterior* likelihood

$$\hat{\mathbf{u}}_{MAP} = \arg\min_{\mathbf{u}} \frac{1}{2\sigma^2}\|\dot{\mathbf{V}} - \mathbf{MKu}\|_2^2 + \lambda\mathbf{u}\cdot\mathbf{n} \qquad \text{with} \quad u_t \geq 0 \quad \forall t, \qquad (6)$$

where $\mathbf{n}$ is a vector of ones and $\lambda$ is the Lagrange multiplier for the regularizer, or equivalently parametrizes the exponential prior distribution over $u(t)$. As mentioned above, this maximum *a posteriori* (MAP) estimate corresponds to a product exponential prior on the synaptic input $u_t$; the multiplier $\lambda$ may be chosen as the expected synaptic input per unit time. It is well known that this type of prior has a sparsening effect, shrinking small values of $u_{ML}(t)$ to zero. This is visible in figure 2 (right); we see that the small, noise-matching synaptic activity is effectively suppressed, permitting much more accurate detection of the true input spike timing.

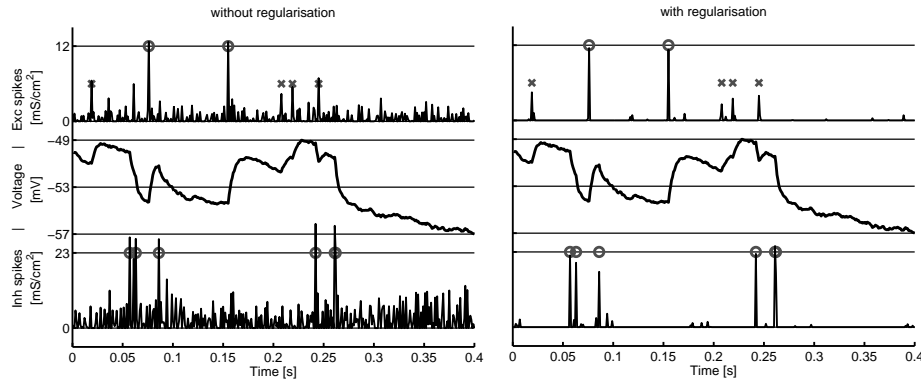

**Figure 2**: Inferring synaptic inputs to a passive membrane. Top traces: excitatory inputs; bottom: inhibitory inputs; middle: the resulting voltage trace. Left panels: synaptic inputs inferred by ML; right: MAP estimates under the exponential (shrinkage) prior. Note the overfitting by the ML estimate (left) and the higher accuracy under the MAP estimate (right); in particular note that the two excitatory synapses of differing magnitudes may easily be distinguished.

### 3.3 Inferring synaptic input and channel distribution in an active model

The optimization is, as mentioned earlier, jointly convex in both channel densities and synaptic input. We illustrate the simultaneous inference of channel densities and synaptic inputs in a single compartment, writing the model as:

$$\frac{dV}{dt} = \sum_{c=1}^{N_{chan}} \bar{g}_c g_c(V, t)(V_c - V(t)) + \sum_{s=1}^{S} g_s(t)(V_s - V(t)) + \sigma dN(t), \qquad (7)$$

with the same channels and synapse types as above. The combination of leak conductance and inhibitory synaptic input leads to very small eigenvalues in $\mathbf{A}$ and slow convergence

when applying the above decomposition; thus, to speed convergence here we coarsened the time resolution of the synaptic input from 0.1 ms to 0.2 ms. Figure 3 demonstrates the accuracy of the results.

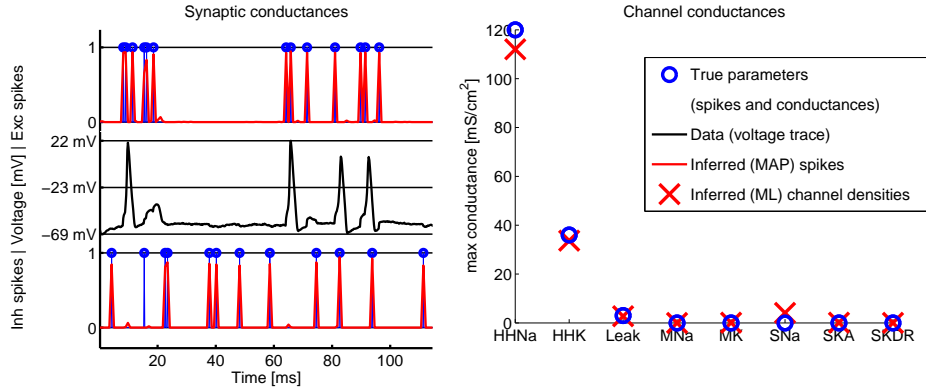

**Figure 3**: Joint inference of synaptic input and channel densities. The true parameters are in blue, the inferred parameters in red. The top left panel shows the excitatory synaptic input, the middle left panel the voltage trace (the only data) and the bottom left traces the inhibitory synaptic input. The right panel shows the true and inferred channel densities; channels are the same as in 3.1.

### 3.4 Eigenvector analysis for a single-compartment model

Finally, as discussed above, the eigenvectors ("principal components") of the loglikelihood Hessian $\mathbf{A}$ carry significant information about the dependence and redundancy of the parameters under study here. An example is given in figure 4; for simplicity, we restrict our attention again to the single-compartment case. In the leftmost panels, we see that the direction $\mathbf{a}_{most}$ most highly-constrained by the data – the eigenvector corresponding to the largest eigenvalue of $\mathbf{A}$ – turns out to have the intuitive form of the balance between $Na^+$ and $K^+$ channels. When we perturb this balance slightly (that is, when we shift the model parameters slightly along this direction in parameter space, $\mathbf{a}_{ML} \rightarrow \mathbf{a}_{ML} + \epsilon \mathbf{a}_{most}$), the cell's behavior changes dramatically. Conversely, the least-sensitive direction, $\mathbf{a}_{least}$, corresponds roughly to the balance between the concentrations of two $Na^+$ channels with similar kinetics, and moving in this direction in parameter space ($\mathbf{a}_{ML} \rightarrow \mathbf{a}_{ML} + \epsilon \mathbf{a}_{least}$) has a negligible effect on the model's dynamical behavior.

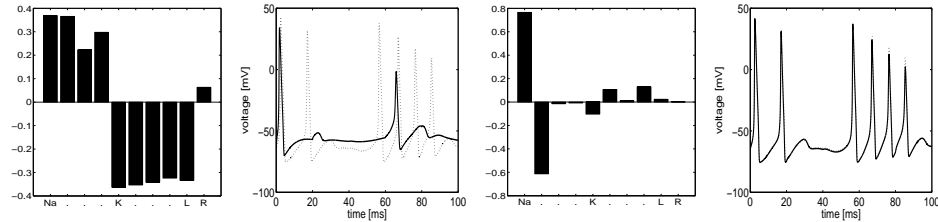

**Figure 4**: Eigenvectors of $\mathbf{A}$ corresponding to largest ($\mathbf{a}_{most}$, left) and smallest ($\mathbf{a}_{least}$, right) eigenvalues, and voltage traces of the model neuron after equal sized perturbations by both (solid line: perturbed model; dotted line: original model). The first four parameters are the concentrations of four $Na^+$ channels (the first two of which are in fact the same Hodgkin-Huxley channel, but with slightly different kinetic parameters); the next four of $K^+$ channels; the next of the leak channel; the last of $1/C$.

## 4 Discussion and future work

We have developed a probabilistic regression framework for estimation of biophysical single neuron properties and synaptic input. This framework leads directly to efficient, globally-convergent algorithms for determining these parameters, and also to well-founded methods for analyzing the uncertainty of the estimates. We believe this is a key first step towards applying these techniques in detailed, quantitative studies of dendritic input and processing *in vitro* and *in vivo*. However, some important caveats – and directions for necessary future work – should be emphasized.

**Observation noise:** While we have explicitly allowed current noise in our main evolution equation (1) (and experimented with a variety of other current- and conductance-noise terms; data not shown), we have assumed that the resulting voltage $V(t)$ is observed noiselessly, with sufficiently high sampling rates. This is a reasonable assumption when voltage is recorded directly, via patch-clamp methods. However, while voltage-sensitive imaging techniques have seen dramatic improvements over the last few years (and will continue to do so in the near future), currently these methods still suffer from relatively low signal-to-noise ratios and spatiotemporal sampling rates. While the procedure proved to be robust to low-level noise of various forms (data not shown), it will be important to relax the noiseless-observation assumption, most likely by adapting standard techniques from the hidden Markov model signal processing literature [11].

**Hidden branches:** Current imaging and dye technologies allow for the monitoring of only a fraction of a dendritic tree; therefore our focus will be on estimating the properties of these sub-structures. Furthermore, these dyes diffuse very slowly and may miss small branches of dendrites, thereby effectively creating unobserved current sources.

**Misspecified channel kinetics and channels with chemical dependence:** Channels dependent on unobserved variables (e.g., $Ca^{++}$-dependent $K^+$ channels), have not been included in the model. The techniques described here may thus be applied unmodified to experimental data for which such channels have been blocked pharmacologically. However, we should note that our methods extend directly to the case where simultaneous access to voltage and calcium signals is possible; more generally, one could develop a semi-realistic model of calcium concentration, and optimize over the parameters of this model as well. We have discussed in some detail (e.g. figure 1) the effect of misspecifications of voltage-dependent channel kinetics and how the most relevant channels may be selected by supplying sufficiently rich "channel libraries". Such libraries can also contain several "copies" of the same channel, with one or more systematically varying parameters, thus allowing for a limited search in the nonlinear space of channel kinetics. Finally, in our discussion of "equivalence classes" of channels (figure 4), we illustrate how eigenvector analysis of our objective function allows for insights into the joint behaviour of channels.

## Footnotes

[1]Note that $C_x$ is the proportionality constant between the externally injected electrode current and $\frac{dV}{dt}$. It is linear in the data and can be included with the other parameters $a_i$ in the joint estimation.

[2]In the case that the reversal potentials $E_j$ are unknown as well, we may estimate these terms by separating the term $\bar{g}_j g_j(t)(V(t) - E_j)$ into $\bar{g}_j g_j(t)V(t)$ and $(\bar{g}_j E_j)g_j(t)$, thereby increasing the number of parameters in the regression by one per channel; $E_j$ is then set to $(\bar{g}_j E_j)/\bar{g}_j$.

## References

[1] Jolivet, Lewis, and Gerstner, 2004. J. Neurophysiol., 92, 959-976.

[2] Vanier and Bower, 1999. J. Comput. Neurosci., 7(2), 149-171.

[3] Goldman, Golowasch, Marder and Abbott, 2001. J. Neurosci., 21(14), 5229-5238.

[4] Wood, Gurney and Wilson, 2004. Neurocomputing, 58-60, 1109-1116.

[5] Morse, Davison and Hines, 2001. Soc. Neurosci. Abs., 606.5.

[6] Baldi, Vanier and Bower, 1998. J. Comp. Neurosci., 5(3), 285-314.

[7] Press et al., 1992. Numerical Recipes in C, CUP.

[8] Hodgkin and Huxley, 1952. J. Physiol., 117.

[9] Poirazi, Brannon and Mel, 2003. Neuron, 37(6), 977-87.

[10] Mainen, Joerges, Huguenard, and Sejnowski, 1995. Neuron, 15(6), 1427-39.

[11] Rabiner, 1989. Proc. IEEE, 77(2), 257-286.
